# Recognizing Hand–Printed Letters and Digits

**Gale L. Martin    James A. Pittman**
MCC, Austin, Texas  78759

## ABSTRACT

We are developing a hand–printed character recognition system using a multi-layered neural net trained through backpropagation. We report on results of training nets with samples of hand-printed digits scanned off of bank checks and hand–printed letters interactively entered into a computer through a stylus digitizer. Given a large training set, and a net with sufficient capacity to achieve high performance on the training set, nets typically achieved error rates of 4–5% at a 0% reject rate and 1–2% at a 10% reject rate. The topology and capacity of the system, as measured by the number of connections in the net, have surprisingly little effect on generalization. For those developing practical pattern recognition systems, these results suggest that a large and representative training sample may be the single, most important factor in achieving high recognition accuracy. From a scientific standpoint, these results raise doubts about the relevance to backpropagation of learning models that estimate the likelihood of high generalization from estimates of capacity. Reducing capacity does have other benefits however, especially when the reduction is accomplished by using local receptive fields with shared weights. In this latter case, we find the net evolves feature detectors resembling those in visual cortex and Linsker's orientation–selective nodes.

Practical interest in hand–printed character recognition is fueled by two current technology trends: one toward systems that interpret hand–printing on hard–copy documents and one toward notebook–like computers that replace the keyboard with a stylus digitizer. The stylus enables users to write and draw directly on a flat panel display. In this paper, we report on results applying multi–layered neural nets trained through backpropagation (Rumelhart, Hinton, & Williams, 1986) to both cases.

Developing pattern recognition systems is typically a two–stage process. First, intuition and experimentation are used to select a set of features to represent the raw input pattern. Then a variety of well–developed techniques are used to optimize the classifier system that assumes this featural representation. Most applications of backpropagation learning to character recognition use the learning capabilities only for this latter

stage—developing the classifier system (Burr, 1986; Denker, Gardner, Graf, Henderson, Howard, Hubbard, Jackel, Baird, & Guyon, 1989; Mori & Yokosawa, 1989; Weideman, Manry, & Yau, 1989). However, backpropagation learning affords the opportunity to optimize feature selection and pattern classification simultaneously . We avoid using pre-determined features as input to the net in favor of using a pre-segmented, size-normalized grayscale array for each character. This is a first step toward the goal of approximating the raw input projected onto the human retina, in that no pre-processing of the input is required.

We report on results for both hand-printed digits and letters. The hand-printed digits come from a set of 40,000 hand-printed digits scanned from the numeric amount region of "real-world" bank checks. They were pre-segmented and size-normalized to a 15x24 grayscale array. The test set consists of 4,000 samples and training sets varied from 100 to 35,200 samples. Although it is always difficult to compare recognition rates arising from different pattern sets, some appreciation for the difficulty of categorization can be gained using human performance data as a benchmark. An independent person categorizing the test set of pre-segmented, size-normalized digits achieved an error rate of 3.4%. This figure is considerably below the near-perfect performance of operators keying in numbers directly from bank checks, because the segmentation algorithm is flawed.

Working with letters, as well as digits, enables tests of the generality of results on a different pattern set having more than double the number of output categories. The hand-printed letters come from a set of 8,600 upper-case letters collected from over 110 people writing with a stylus input device on a flat panel display. The stylus collects a sequence of x-y coordinates at 200 points per second at a spatial resolution of 1000 points per inch. The temporal sequence for each character is first converted to a size-normalized bitmap array, keeping aspect ratio constant. We have found that recognition accuracy is significantly improved if these bitmaps are blurred through convolution with a gaussian distribution. Each pattern is represented as a 15x24 grayscale image. A test set of 2,368 samples was extracted by selecting samples from 18 people, so that training sets were generated by people different from those generating the test set. Training set sizes ranged from 500 to roughly 6,300 samples.

## 1   HIGH RECOGNITION  ACCURACY

We find relatively high recognition accuracy for both pattern sets. Table 1[1] reports the minimal error rates achieved on the test samples for both pattern sets, at various reject rates. In the case of the hand-printed digits, the 4% error rate (0% rejects) ap-

---

1.     Effects of the number of training samples and network capacity and topology are reported in the next section. Nets were trained to error rates of 2–3%. Training began with a learning rate of .05 and a momentum value of .9. The learning rate was decreased when training accuracy began to oscillate or had stabilized for a large number of training epochs. We evaluate the output vector on a winner-take-all basis, as this consistently improves accuracy and results in network parameters having a smaller effect on performance.

proaches the 3.4% errors made by the human judge. This suggests that further improvements to generalization will require improving segmentation accuracy. The fact that an error rate of 5% was achieved for letters is promising. Accuracy is fairly high,

**Table 1:** Error rates of best nets trained on largest sample sets and tested on new samples

| REJECT RATE | DIGITS | LETTERS |
|---|---|---|
| 0% | 4% | 5% |
| 5% | 3% | 3% |
| 10% | 1% | 2% |
| 35% | .001% | .003% |

even though there are a large number of categories (26). This error rate may be adequate for applications where contextual constraints can be used to significantly boost accuracy at the word-level.

## 2 MINIMAL NETWORK CAPACITY AND TOPOLOGY EFFECTS

The effects of network parameters on generalization have both practical and scientific significance. The practical developer of pattern recognition systems is interested in such effects to determine whether limited resources should be spent on trying to optimize network parameters or on collecting a larger, more representative training set. For the scientist, effects of capacity bear on the relevance of learning models to backpropagation.

A central premise of most general models of learning-by-example is that the size of the initial search space—the capacity of the system—determines the number of training samples needed to achieve high generalization performance. Learning is conceptualized as a search for a function that maps all possible inputs to their correct outputs. Learning occurs by comparing successive samples of input-output pairs to functions in a search space. Functions inconsistent with training samples are rejected. Very large training sets narrow the search down to a function that closely approximates the desired function and yields high generalization. The capacity of a learning system—the number of functions it can represent—determines generalization, since a larger initial search space requires more training samples to narrow the search sufficiently . This suggests that to improve generalization, capacity should be minimized. Unfortunately, it is typically unclear how to minimize capacity without eliminating the desired function from the search space. A heuristic, which is often suggested, is that *simple is usually better.* It receives support from experience in curve fitting. Low-order polynomials typically extrapolate and interpolate better than high-order polynomials (Duda & Hart, 1973).

Extensions of the heuristic to neural net learning propose reducing capacity by reducing the number of connections or the number of bits used to represent each connection

weight (Baum & Haussler, 1989; Denker, Schwartz, Wittner, Solla, Howard, Jackel, & Hopfield, 1987). We manipulated the capacity of nets in a number of ways: 1) varying the number of hidden nodes, 2) limiting connectivity between layers so that nodes received input from only local areas, and 3) sharing connection weights between hidden nodes. We found only negligible effects on generalization.

## 2.1 NUMBER OF HIDDEN NODES

Figure 1 presents generalization results as a function of training set size for nets having one hidden layer and varying numbers of hidden nodes. The number of free parameters (i.e., number of connections and biases) in each case is presented in parentheses. Despite considerable variation in the number of free parameters, using nets with fewer hidden nodes did not improve generalization.

Baum & Haussler (1989) estimate the number of training samples required to achieve an error rate $e$ (where $0 < e \leq 1/8$) on the generalization test, when an error rate of $e/2$ has been achieved on the training set. They assume a feed–forward net with one hidden layer and $W$ connections. The estimates are distribution–free in the sense that calculations assume an arbitrary to–be–learned function. If the number of training samples is of order $\frac{W}{\epsilon} \log \frac{N}{\epsilon}$, where N refers to the number of nodes, then it is a near certainty that the net will achieve generalization rates of $(1 - e)$. This estimate is the upper–bound on the number of training samples needed. They also provide a lower

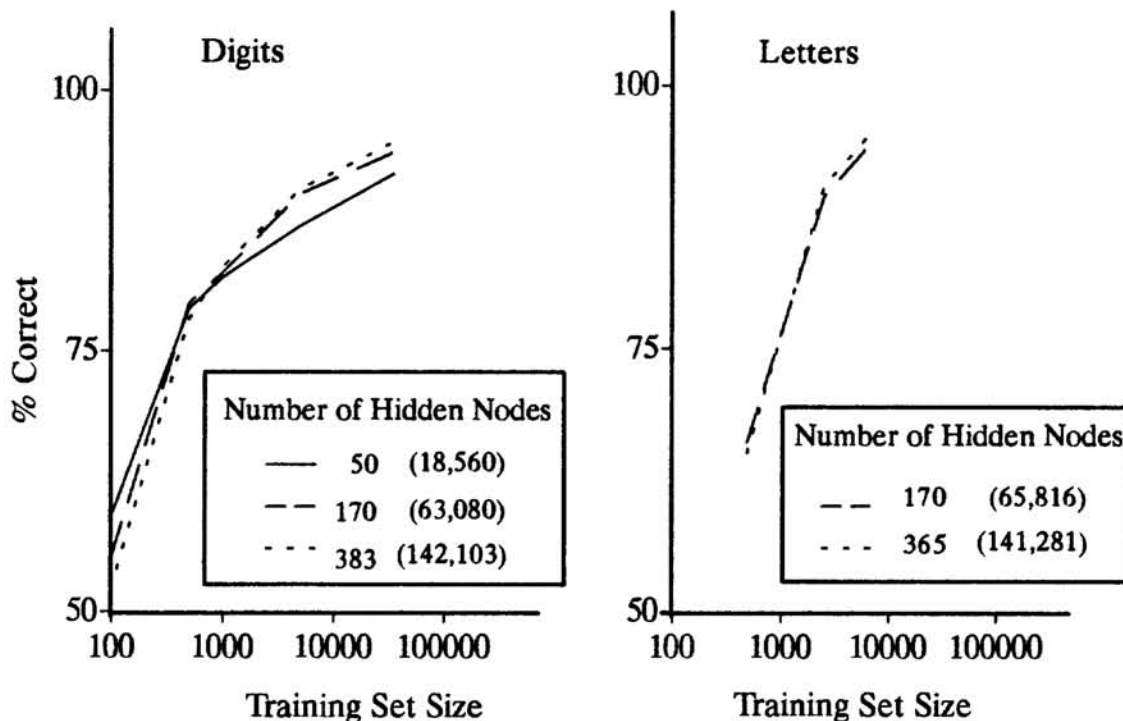

Figure 1. Effect of number of hidden nodes and training set size on generalization.

bound estimate, on the order of $W/e$. Using fewer than this number of samples will, for some functions, fail to achieve generalization rates of $(1 - e)$. The fact that we find no advantage to reducing the number of connections conflicts with Baum & Haussler's estimates and the underlying assumption that capacity plays a strong role in determining generalization.

Baum & Haussler also suggest using a constant of proportionality of 1 in their estimates. This implies that achieving error rates of 10% or less on new samples requires about 10 times as many training examples as there are connection weights in the net. For our largest nets, this implies a requirement of roughly a million training samples, which most developers would regard as prohibitively large. We found that about 5,000 samples were sufficient. Thus, a *sufficiently large* training sample does not imply a *prohibitively large* sample, at least for character recognition. We find that sample sizes of the order of thousands to tens of thousands yield performance very close to human levels. One reason for the discrepancy is that Baum & Haussler's estimates are distribution–free in the sense that they reflect worst-case scenarios across all possible functions the net might learn. Presumably, the functions underlying most natural pattern recognition tasks are not representative of the set of all possible functions. These results raise doubts about the relevance to natural pattern recognition of learning models based on worst–case analyses, because content may greatly impact generalization.

## 2.2 LOCAL CONNECTIVITY AND SHARED WEIGHTS

A more biologically plausible way to reduce capacity is to limit connectivity between layers to local areas and to use shared weights. For example, visual cortex contains neurons, each of which is responsive to a feature such as an oriented line appearing in a small, local region on the retina (Hubel & Wiesel, 1979). A given oriented line-detector is essentially replicated across the visual field, so that the same feature can be detected wherever it appears. In this sense, the connections feeding into an oriented-line detector are shared across all similar line-detectors for different areas of the visual field. In an artificial neural net, local structure is achieved by limiting connectivity. A given hidden node receives input from only local areas in the input or hidden layer preceding it. Weight sharing is achieved by linking the incoming weights across a set of hidden nodes. Corresponding weights leading into these nodes are randomly initialized to the same values and forced to have equivalent updates during learning. In this way the net evolves the same local feature set that is invariant across the input array. Several demonstrations exist indicating that local connectivity and shared weights improve generalization performance in tasks where position invariance is required (le Cun, 1989; Rumelhart, Hinton, & Williams, 1986).

We examined the benefits of using local receptive fields with shared weights for hand-printed character recognition, where position invariance was not required. This does not minimize the importance of position invariance. However, it is only one of many necessary invariants underlying reliable pattern recognition. Unfortunately, most of these invariants have not been explicitly specified. So we don't know how to bias a net toward discovering them. Testing the role of local receptive fields with shared weights

in situations where position invariance is not required is relevant to discovering whether these constraints have a role other than in promoting position invariance.

As indicated in Figure 2, we find only slightly improved generalization in moving from nets with global connectivity between layers to nets with local receptive fields or to nets with local receptive fields and shared weights. This is true despite the fact that the number of free parameters is substantially reduced. The positive effects that do occur are at relatively small training set sizes. This may explain why others have reported a greater degree of improved generalization by using local receptive fields (Honavar & Uhr, 1988). The data reported are for networks with two hidden layers. *Global* nets had 150 nodes in the first layer and 50 nodes in the second. In the *Local* nets, first hidden layer nodes (540) received input from 5x8 local and overlapping regions (offset by 2 pixels) on the input array. Second hidden layer nodes (100) and output layer nodes had global receptive fields. The *Local, Shared* nets had 540 nodes in the first hidden layer with shared weights and, at the second hidden layer, either 102 (digits) or 180 (letters) nodes with local, overlapping, and shared receptive fields of size 4x6 on the 1st

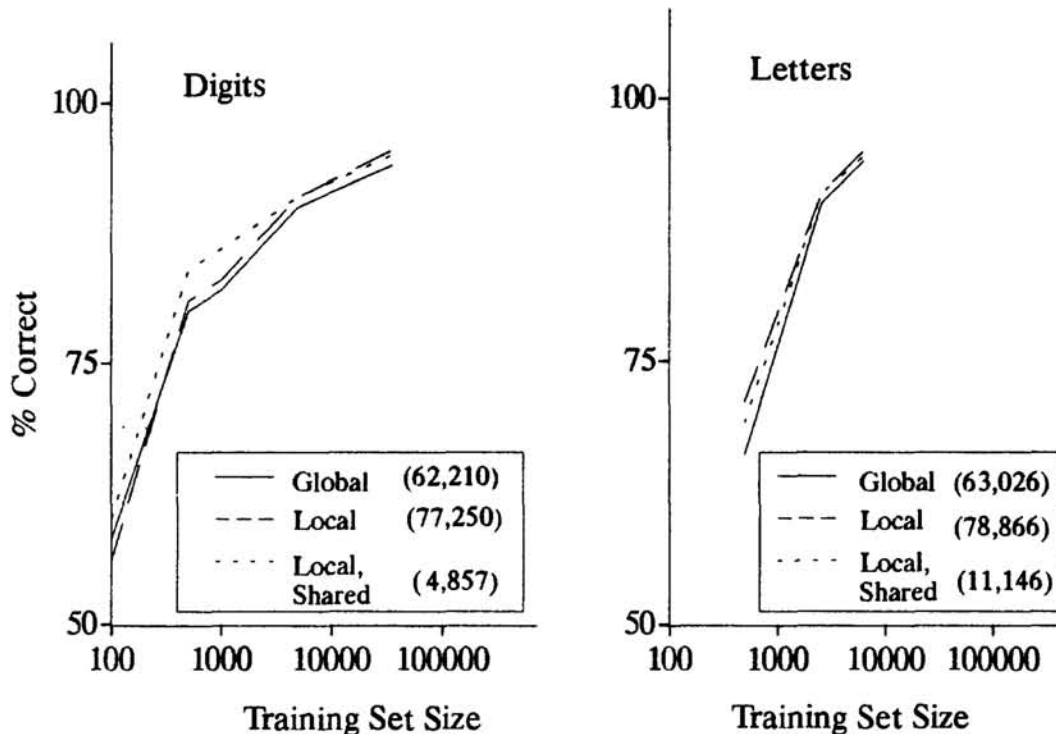

Figure 2.    Effects of net capacity and topology on generalization.

hidden layer. We have experimented with a large variety of different net architectures of this sort, varying the number of hidden nodes, the sizes and overlap of local receptive fields, and the use of local receptive fields with and without shared weights in one or both hidden layers. The fact that we've found little difference in generalization for two different pattern sets across such variations in network architectures argues for the generality of the results.

## 2.3    DISCUSSION

Given an architecture that enables relatively high training performance, we find only small effects of network capacity and topology on generalization performance. A large training set yields relatively high recognition accuracy in a robust way across most net architectures with which we've worked. These results suggest some practical advice to those developing hand–printed character recognition systems. If optimizing generalization performance is the goal, it is probably better to devote limited resources to collecting a very large, representative training set than to extensive experimentation with different net architectures. The variations in net capacity and topology we've examined do not substantially affect generalization performance for sufficiently large training sets. *Sufficiently large* should be interpreted as on the order of a thousand to tens of thousands of samples for hand–printed character recognition.

From a theoretical standpoint, the negligible effects of network capacity on generalization performance contradicts the central premise of machine learning that the size of the initial hypothesis space determines learning performance. This challenges the relevance, to backpropagation learning, of statistical models that estimate likelihood of high generalization performance from estimates of capacity. Due to the gradient descent nature of backpropagation learning, not all functions that can be represented will be visited during learning. The negligible effects of capacity suggest that the number of functions visited during learning constitutes only a very small percentage of the total possible functions that can be represented.

There are a number of reasons for believing that capacity might impact generalization performance in other circumstances. We regularly train only to 2–3% error rates. This helps to avoid the possibility of overfitting the data, although we have seen no indication of this when we have trained to higher levels, as long as we use large training sets. It is also possible that the number of connections is not a good measure of capacity. For example, the amount of information that can be passed on by a given connection may be a better measure than the number of connections. At this conference, le Cun, Denker, Solla, Howard, & Jackel have also presented evidence that removing unimportant weights from a network may be a better way to reduce capacity. However, the fact that generalization rates come very close to human accuracy levels, even for nets with extremely large numbers of free parameters, suggests that general effects of net capacity and topology are, at best, small in comparison to effects of training set size. We don't deny that there are likely to be net topologies that push performance up to human accuracy levels, presumably by biasing the net toward discovering the range of invariants that underlie human pattern recognition. The problem is that only a few of these invariants have been explicitly specified (e.g., position, size, rotation), and so it is not possible to bias a net toward discovering the full range.

## 3  ADVANTAGES OF REDUCING  CAPACITY

Although reducing gross indicators of capacity may not significantly improve generalization, there are good practical and scientific reasons for doing it.  A good reason to reduce the number of connections is to speed processing.  Also, using local receptive fields with shared weights biases a net toward position invariance, and toward producing a simpler, more modular internal representation which can be replicated across a large retina.  This has important implications for developing nets that combine character segmentation with recognition.

Using local receptive fields with shared weights also offers promise for increasing our understanding of how the net correctly classifies patterns because the number of distinct receptive fields is greatly reduced.  Figure 3 depicts Hinton diagrams of local re-

Digits                    Letters

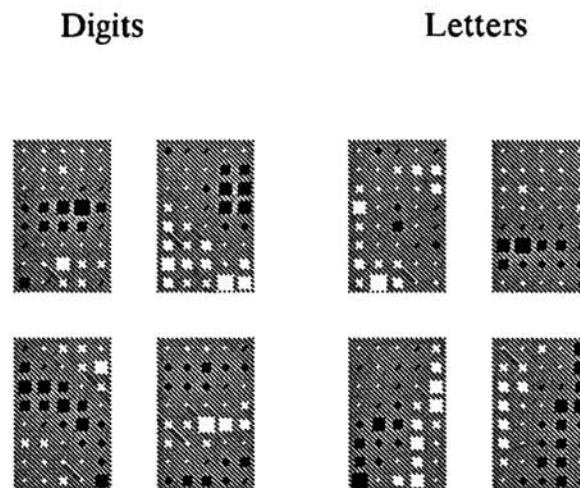

Figure 3.  Receptive fields that evolved in 1st hidden layer nodes in nets with
local receptive fields having shared weights.

ceptive fields from 1st hidden layer nodes in nets with shared weights trained on digits or letters.  Each of the eight large, gray rectangles corresponds to the receptive field for a hidden node.  The four on the left came from a net trained on digits; those on the right from a net trained on letters. Within each of these eight, the black rectangles correspond to negative weights and the white to positive weights. The size of the black and white rectangles reflects the magnitude of the weights.

The local feature detectors that develop for both pattern sets appear to be oriented line and edge detectors.  These are similar to oriented line and edge detectors found in visual cortex (Hubel & Wiesel, 1979) and to Linsker's (1986,1988) orientation-selective nodes, which emerge from a self-adaptive net exposed to random patterns.  In Linsker's case, the feature detectors develop as an emergent property of the principle that the signal transformation occurring from one layer to the next should maximize the information that output signals convey about input signals.  The fact that similar

feature detectors emerge in backpropagation nets trained on "natural" patterns is interesting because there were no explicit constraints to maximize information flow between layers in the backpropagation nets and because categorization is typically viewed as an abstraction process involving considerable loss of category-irrelevant information.

## References

Baum, E. and Haussler, D. (1989) What size net gives valid generalization? in D. S. Touretzky (Ed.) *Advances in neural information processing systems I*, Morgan Kaufman.

Burr, D. J. (1986) A neural network digit recognizer. *Proceedings of the 1986 International conference on systems, man and cybernetics*, Atlanta, Georgia. pp. 1621–1625.

Denker, J. S., Gardner, W. R., Graf, H. P., Henderson, D., Howard, R. E., Hubbard, W., Jackel, L. D., Baird, H. S., and Guyon, I. (1989) Neural network recognizer for hand-written zip code digits in D. S. Touretzky (Ed.) *Advances in neural information processing systems I*, Morgan Kaufman.

Denker, J. S., Schwartz, D., Wittner, B., Solla, S., Howard, R. E., Jackel, L. D., & Hopfield, J. (1987) Large automatic learning, rule extraction and generalization. *Complex Systems, 1*, pp. 877–933.

Duda, R. O., and Hart, P. E. (1973) *Pattern classification and scene analysis.* NY: John Wiley & Sons.

Honavar, V. and Uhr, L. (1988) Experimental results indicate that generation, local receptive fields and global convergence improve perceptual learning in connectionist networks. CS-TR 805. Computer Science Department, University of Wisconsin-Madison.

Hubel, D. H. and Wiesel, T. N. (1979) Brain mechanisms of vision. *Scientific American, 241*, pp. 150–162.

le Cun, Y. (1989) Generalization and network design strategies. Technical Report CRG-TR-89-4, Department of Computer Science, University of Toronto.

Linsker, R. (1986) From basic network principles to neural architecture; Emergence of orientation-selective cells. *Proceedings of the National Academy of Sciences, USA, 83*, pp. 8390–8394.

Linsker, R. (1988) Towards an organizing principle for a layered perceptual network in D. Anderson (Ed.) *Neural information processing systems.* American Institute of Physics.

Mori, Y. and Yokosawa, K. (1989) Neural networks that learn to discriminate similar kanji characters. in . D. S. Touretzky (Ed.) *Advances in neural information processing systems I*, Morgan Kaufman.

Rumelhart, D. E., Hinton, G. E., & Williams, R. J. Learning internal representations by error propagation  in D. E. Rumelhart & J. L. McClelland (Editors) *Parallel distributed processing: V. 1*. Cambridge, Mass.: MIT Press, 1986

Weideman, W. E., Manry, M T. & Yau, H. C. (1989) A comparison of a nearest neighbor classifier and a neural network for numeric handprint character recognition. IEEE International Conference on Neural Networks, Washington, D. C., 1989.

**Acknowledgements**

We would like to thank the NCR corporation for loaning us the set of hand-printed digits and Joyce Conner, Janet Kilgore, and Kay Bauer for their invaluable help in collecting the set of hand-printed letters.
